# Information Measure Based Skeletonisation

**Sowmya Ramachandran**
Department of Computer Science
University of Texas at Austin
Austin, TX 78712-1188

**Lorien Y. Pratt** *
Department of Computer Science
Rutgers University
New Brunswick, NJ 08903

## Abstract

Automatic determination of proper neural network topology by trimming over-sized networks is an important area of study, which has previously been addressed using a variety of techniques. In this paper, we present Information Measure Based Skeletonisation (IMBS), a new approach to this problem where *superfluous* hidden units are removed based on their *information measure* (IM). This measure, borrowed from decision tree induction techniques, reflects the degree to which the hyperplane formed by a hidden unit discriminates between training data classes. We show the results of applying IMBS to three classification tasks and demonstrate that it removes a substantial number of hidden units without significantly affecting network performance.

## 1   INTRODUCTION

Neural networks can be evaluated based on their learning speed, the space and time complexity of the learned network, and generalisation performance. Pruning over-sized networks (skeletonisation) has the potential to improve networks along these dimensions as follows:

- Learning Speed: Empirical observation indicates that networks which have been constrained to have fewer parameters lack flexibility during search, and so tend to learn slower. Training a network that is larger than necessary and

*This work was partially supported by DOE #DE-FG02-91ER61129, through subcontract #097P753 from the University of Wisconsin.

trimming it back to a reduced architecture could lead to improved learning speed.

- Network Complexity: Skeletonisation improves both space and time complexity by reducing the number of weights and hidden units.

- Generalisation: Skeletonisation could constrain networks to generalise better by reducing the number of parameters used to fit the data.

Various techniques have been proposed for skeletonisation. One approach [Hanson and Pratt, 1989, Chauvin, 1989, Weigend *et al.*, 1991] is to add a cost term or bias to the objective function. This causes weights to decay to zero unless they are reinforced. Another technique is to measure the increase in error caused by removing a parameter or a unit, as in [Mozer and Smolensky, 1989, Le Cun *et al.*, 1990]. Parameters that have the least effect on the error may be pruned from the network.

In this paper, we present Information Measure Based Skeletonisation (IMBS), an alternate approach to this problem, in which *superfluous* hidden units in a single hidden-layer network are removed based on their *information measure* (IM). This idea is somewhat related to that presented in [Siestma and Dow, 1991], though we use a different algorithm for detecting superfluous hidden units.

We also demonstrate that when IMBS is applied to a vowel recognition task, to a subset of the Peterson-Barney 10-vowel classification problem, and to a heart disease diagnosis problem, it removes a substantial number of hidden units without significantly affecting network performance.

## 2    IM AND THE HIDDEN LAYER

Several decision tree induction schemes use a particular information-theoretic measure, called IM, of the degree to which an attribute separates (discriminates between the classes of) a given set of training data [Quinlan, 1986]. IM is a measure of the information gained by knowing the value of an attribute for the purpose of classification. The higher the IM of an attribute, the greater the uniformity of class data in the subsets of feature space it creates.

A useful simplification of the sigmoidal activation function used in back-propagation networks [Rumelhart *et al.*, 1986] is to reduce this function to a threshold by mapping activations greater than 0.5 to 1 and less than 0.5 to 0. In this simplified model, the hidden units form hyperplanes in the feature space which separate data. Thus, they can be considered analogous to binary-valued attributes, and the IM of each hidden unit can be calculated as in decision tree induction [Quinlan, 1986].

Figure 1 shows the training data for a fabricated two-feature, two-class problem and a possible configuration of the hyperplanes formed by each hidden unit at the end of training. Hyperplane $h1$'s higher IM corresponds to the fact that it separates the two classes better than $h2$.

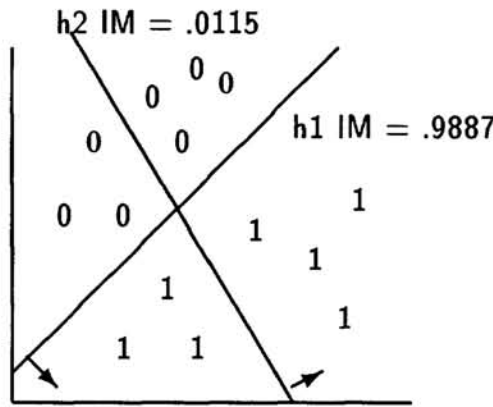

Figure 1: Hyperplanes and their IM. Arrows indicate regions where hidden units have activations > 0.5.

## 3   IM TO DETECT SUPERFLUOUS HIDDEN UNITS

One of the important goals of training is to adjust the set of hyperplanes formed by the hidden layer so that they separate the training data.[1] We define *superfluous* units as those whose corresponding hyperplanes are not necessary for the proper separation of training data. For example, in Figure 1, hyperplane $h2$ is superfluous because:

1. $h1$ separates the data better than $h2$ and

2. $h2$ does not separate the data in either of the two regions created by $h1$.

The IMBS algorithm to identify superfluous hidden units, shown in Figure 2, recursively finds hidden units that are necessary to separate the data and classifies the rest as superfluous. It is similar to the decision tree induction algorithm in [Quinlan, 1986].

The hidden layer is skeletonised by removing the superfluous hidden units. Since the removal of these units perturbs the inputs to the output layer, the network will have to be trained further after skeletonisation to recover lost performance.

## 4   RESULTS

We have tested IMBS on three classification problems, as follows:

1. Train a network to an acceptable level of performance.

2. Identify and remove superfluous hidden units.

3. Train the skeletonised network further to an acceptable level of performance.

We will refer to the stopping point of training at step 1 as the *skeletonisation point (SP)*; further training will be referred to in terms of SP + number of training epochs.

**Input:**
  *Training data*
  *Hidden unit activations for each training data pattern.*
**Output:**
  *List of superfluous hidden units.*
**Method:**
  *main  ident-superfluous-hu*
  *begin*
    *data-set← training data*
    *useful-hu-list← nil*
    *pick-best-hu(data-set,useful-hu-list)*
    *output hidden units that are not in useful-hu-list*
  *end*
  *procedure pick-best-hu(data-set, useful-hu-list)*
  *begin*
    *if all the data in data-set belong to the same class then return*
    *Calculate IM of each hidden unit.*
    *h1← hidden unit with best IM.*
    *add h1 to the useful-hu list*
    *ds1← all the data in data-set for which h1 has an activation of > .5*
    *ds2← all the data in data-set for which h1 has an activation of <= .5*
    *pick-best-hu(ds1, useful-hu-list)*
    *pick-best-hu(ds2, useful-hu-list)*
  *end*

Figure 2: IMBS: An Algorithm for Identifying Superfluous Hidden Units

For each problem, data was divided into a training set and a test set. Several networks were run for a few epochs with different back-propagation parameters $\eta$ (learning rate) and $\alpha$ (momentum) to determine their locally optimal values.

For each problem, we chose an initial architecture and trained 10 networks with different random initial weights for the same number of epochs. The performances of the original (i.e. the network before skeletonisation) and the skeletonised networks, measured as number of correct classifications of the training and test sets, was measured both at SP and after further training. The retrained skeletonised network was compared with the original network at SP as well as the original network that had been trained further for the same number of weight updates.[2] All training was via the standard back-propagation algorithm with a sigmoidal activation function and updates after every pattern presentation [Rumelhart *et al.*, 1986]. A paired T-test [Siegel, 1988] was used to measure the significance of the difference in performance between the skeletonised and original networks. Our experimental results are summarised in Figure 3, and Tables 1 and 2; detailed experimental conditions are given below.

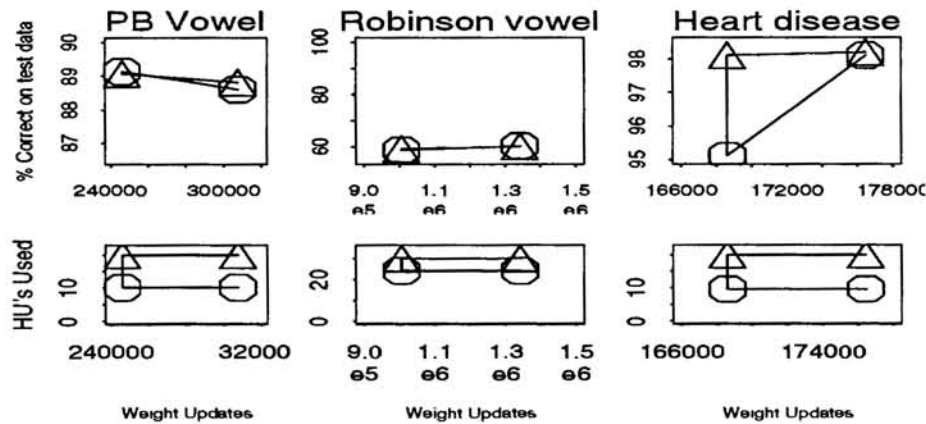

Figure 3: Summary of experimental results. Circles represent skeletonised networks; triangles represent unskeletonised networks for comparison. Note that when performance drops upon skeletonisation, the original performance level is recovered within a few weight updates. In all cases, hidden unit count is reduced.

## 4.1   PETERSON-BARNEY DATA

IMBS was first evaluated on a 3-class subset of the Peterson-Barney 10-vowel classification data set, originally described in [Peterson and Barney, 1952], and recreated by [Watrous, 1991]. This data consists of the formant values F1 and F2 for each of two repetitions of each of ten vowels by 76 speaker (1520 utterances). The vowels were pronounced in isolated words consisting of the consonant "h", followed by a vowel, followed by "d". This set was randomly divided into a $\frac{2}{3}, \frac{1}{3}$ training/test split, with 298 and 150 patterns, respectively.

Our initial architecture was a fully connected network with 2 input units, one hidden layer with 20 units, and 3 output units. We trained the networks with $\eta = 1.0$ and $\alpha = 0.001$ until the TSS (total sum of squared error) scores seemed to reach a plateau. The networks were trained for 2000 epochs and then skeletonised.

The skeletonisation procedure removed an average of 10.1 (50.5%) hidden units. Though the average performance of the skeletonised networks was worse than that of the original, this difference was not statistically significant ($p = 0.001$).

## 4.2   ROBINSON VOWEL RECOGNITION

Using data from [Robinson, 1989], we trained networks to perform speaker independent recognition of the 11 steady-state vowels of British English using a training set of LPC-derived log area ratios. Training and test sets were as used by [Robinson, 1989], with 528 and 462 patterns, respectively.

The initial network architecture was fully connected, with 10 input units, 11 output units, and 30 hidden units. Networks were trained with $\eta = 1.0$ and $\alpha = 0.01$, until the performance on the training set exceeded 95%. The networks were trained for 1500 epochs and then skeletonised. The skeletonisation procedure removed an average of 5.8 (19.3%) hidden units. The difference in performance was not statistically significant ($p = 0.001$).

Table 1: Performance of unskeletonised networks

|  | correct classifications | |
|---|---|---|
|  | Training set | Test set |
| Peterson-Barney | | |
| SP | 262.90 (88.22%) | 133.61 (89.07%) |
| SP + 500 | 263.28 (88.35%) | 133.20 (88.80%) |
| Vowel Recognition | | |
| SP | 501.60 (95.00%) | 273.69 (59.24%) |
| SP + 500 | 506.99 (96.02%) | 277.80 (60.13%) |
| Heart Disease | | |
| SP | 805.8 (98.27%) | 402.20 (98.10%) |
| SP + 14 | 806.40 (98.34%) | 402.60 (98.20%) |

Table 2: Mean difference in the number of correct classifications between the original and skeletonised networks. Positive differences indicate that the original network did better after further training. The numbers in parentheses indicate the 99.9% confidence intervals for the mean.

| comparison points | | mean difference | |
|---|---|---|---|
| Original | Skeletonised | Training set | Test set |
| Peterson-Barney | | | |
| SP | SP | 3.10 [-0.83, 7.03] | -0.10 [-2.05, 1.84] |
| SP | SP+1010 | -0.1 [-1.76, 1.56] | 0.7 [-0.73, 2.13] |
| SP+500 | SP+1010 | 0.20 [-1.52, 1.91] | 0.30 [-1.30, 1.90] |
| Robinson Vowel | | | |
| SP | SP | 1.70 [ -2.40, 5.80] | 2.40 [ -2.39, 7.19] |
| SP | SP+620 | -8.2 [-20.33, 3.93] | -4.4 [-18.26, 9.46] |
| SP+500 | SP+620 | -0.30 [ -3.15, 2.55] | -0.30 [ -8.36, 7.76] |
| Heart Disease | | | |
| SP | SP | 20.80 [-5.66, 47.26] | 12.20 [ -1.65, 26.05] |
| SP | SP+33 | 0 [-4.28, +4.28] | 0 [-2.85, 2.85] |
| SP+14 | SP+33 | 0.60 [ -4.55, 5.75] | 0.40 [ -3.03, 3.83] |

## 4.3   HEART DISEASE DATA

Using a 14-attribute set of diagnosis information, we trained networks on a heart disease diagnosis problem [Detrano *et al.*, 1989]. Training and test data were chosen randomly in a $\frac{2}{3}, \frac{1}{3}$ split of 820 and 410 patterns, respectively. The initial networks were fully connected, with 25 input units, one hidden layer with 20 units, and 2 output units. The networks were trained with $\alpha = 1.25$ and $\eta = 0.005$. Training was stopped when the TSS scores seemed to reach a plateau. The networks were trained for 300 epochs and then skeletonised.

The skeletonisation procedure removed an average of 9.6 (48%) hidden units. Here, removing superfluous units degraded the performance by an average of 2.5% on the training set and 3.0% on the test set. However, after being trained further for only 30 epochs, the skeletonised networks recovered to do as well as the original networks.

## 5   CONCLUSION AND EXTENSIONS

We have introduced an algorithm, called IMBS, which uses an information measure borrowed from decision tree induction schemes to skeletonise over-sized back-propagation networks. Empirical tests showed that IMBS removed a substantial percentage of hidden units without significantly affecting the network performance.

Potential extensions to this work include:

- Using decision tree reduction schemes to allow for trimming not only superfluous hyperplanes, but also those responsible for overfitting the training data, in an effort to improve generalisation.

- Extending IMBS to better identify superfluous hidden units under conditions of less than 100% performance on the training data.

- Extending IMBS to work for networks with more than one hidden layer.

- Performing more rigorous empirical evaluation.

- Making IMBS less sensitive to the hyperplane-as-threshold assumption. In particular, a model with variable-width hyperplanes (depending on the sigmoidal gain) may be effective.

### Acknowledgements

Our thanks to Haym Hirsh and Tom Lee for insightful comments on earlier drafts of this paper, to Christian Roehr for an update to the IMBS algorithm, and to Vince Sgro, David Lubinsky, David Loewenstern and Jack Mostow for feedback on later drafts. Matthias Pfister, M.D., of University Hospital in Zurich, Switzerland was responsible for collection of the heart disease data. We used software distributed with [McClelland and Rumelhart, 1988] for many of our simulations.

## Footnotes

[1]This again is not strictly true for hidden units with sigmoidal activation, but holds for the approximate model.

[2]This was ensured by adjusting the number of epochs a network was trained after skeletonisation according to the number of hidden units in the network. Thus, a network with 10 hidden units was trained on twice as many epochs as one with 20 hidden units.

# References

[Chauvin, 1989] Chauvin, Y. 1989. A back-propagation algorithm with optimal use of hidden units. In Touretzky, D. S., editor 1989, *Advances in Neural Information Processing Systems 1*. Morgan Kaufmann, San Mateo, CA. 519–526.

[Detrano *et al.*, 1989] Detrano, R.; Janosi, A.; Steinbrunn, W.; Pfisterer, M.; Schmid, J.; Sandhu, S.; Guppy, K.; Lee, S.; and Froelicher, V. 1989. International application of a new probability algorithm for the diagnosis of coronary artery disease. *American Journal of Cardiology* 64:304–310.

[Hanson and Pratt, 1989] Hanson, Stephen José and Pratt, Lorien Y. 1989. Comparing biases for minimal network construction with back-propagation. In Touretzky, D. S., editor 1989, *Advances in Neural Information Processing Systems 1*. Morgan Kaufmann, San Mateo, CA. 177–185.

[Le Cun *et al.*, 1990] Le Cun, Yann; Denker, John; Solla, Sara A.; Howard, Richard E.; and Jackel, Lawrence D. 1990. Optimal brain damage. In Touretzky, D. S., editor 1990, *Advances in Neural Information Processing Systems 2*. Morgan Kaufmann, San Mateo, CA.

[McClelland and Rumelhart, 1988] McClelland, James L. and Rumelhart, David E. 1988. *Explorations in Parallel Distributed Processing: A Handbook of Models, Programs, and Exercises*. Cambridge, MA, The MIT Press.

[Mozer and Smolensky, 1989] Mozer, Michael C. and Smolensky, Paul 1989. Skeletonization: A technique for trimming the fat from a network via relevance assessment. In Touretzky, D. S., editor 1989, *Advances in Neural Information Processing Systems 1*. Morgan Kaufmann, San Mateo, CA. 107–115.

[Peterson and Barney, 1952] Peterson, and Barney, 1952. Control methods used in a study of the vowels. *J. Acoust. Soc. Am.* 24(2):175–184.

[Quinlan, 1986] Quinlan, J. R. 1986. Induction of decision trees. *Machine Learning* 1(1):81–106.

[Robinson, 1989] Robinson, Anthony John 1989. *Dynamic Error Propagation Networks*. Ph.D. Dissertation, Cambridge University, Engineering Department.

[Rumelhart *et al.*, 1986] Rumelhart, D.; Hinton, G.; and Williams, R. 1986. Learning representations by back-propagating errors. *Nature* 323:533–536.

[Siegel, 1988] Siegel, Andrew F. 1988. *Statistics and data analysis: An Introduction*. John Wiley and Sons. chapter 15, 336–339.

[Siestma and Dow, 1991] Siestma, Jocelyn and Dow, Robert J. F. 1991. Creating artificial neural networks that generalize. *Neural Networks* 4:67–79.

[Watrous, 1991] Watrous, Raymond L. 1991. Current status of peterson-barney vowel formant data. *Journal of the Acoustical Society of America* 89(3):2459–60.

[Weigend *et al.*, 1991] Weigend, Andreas S.; Rumelhart, David E.; and Huberman, Bernardo A. 1991. Generalization by weight-elimination with application to forecasting. In Lippmann, R. P.; Moody, J. E.; and Touretzky, D. S., editors 1991, *Advances in Neural Information Processing Systems 3*. Morgan Kaufmann, San Mateo, CA. 875–882.
